# Attractor Network Dynamics Enable Preplay and Rapid Path Planning in Maze–like Environments

**Dane Corneil**
Laboratory of Computational Neuroscience
École Polytechnique Fédérale de Lausanne
CH-1015 Lausanne, Switzerland
`dane.corneil@epfl.ch`

**Wulfram Gerstner**
Laboratory of Computational Neuroscience
École Polytechnique Fédérale de Lausanne
CH-1015 Lausanne, Switzerland
`wulfram.gerstner@epfl.ch`

## Abstract

Rodents navigating in a well–known environment can rapidly learn and revisit observed reward locations, often after a single trial. While the mechanism for rapid path planning is unknown, the CA3 region in the hippocampus plays an important role, and emerging evidence suggests that place cell activity during hippocampal "preplay" periods may trace out future goal–directed trajectories. Here, we show how a particular mapping of space allows for the immediate generation of trajectories between arbitrary start and goal locations in an environment, based only on the mapped representation of the goal. We show that this representation can be implemented in a neural attractor network model, resulting in bump–like activity profiles resembling those of the CA3 region of hippocampus. Neurons tend to locally excite neurons with similar place field centers, while inhibiting other neurons with distant place field centers, such that stable bumps of activity can form at arbitrary locations in the environment. The network is initialized to represent a point in the environment, then weakly stimulated with an input corresponding to an arbitrary goal location. We show that the resulting activity can be interpreted as a gradient ascent on the value function induced by a reward at the goal location. Indeed, in networks with large place fields, we show that the network properties cause the bump to move smoothly from its initial location to the goal, around obstacles or walls. Our results illustrate that an attractor network with hippocampal–like attributes may be important for rapid path planning.

## 1 Introduction

While early human case studies revealed the importance of the hippocampus in episodic memory [1, 2], the discovery of "place cells" in rats [3] established its role for spatial representation. Recent results have further suggested that, along with these functions, the hippocampus is involved in active spatial planning: experiments in "one–shot learning" have revealed the critical role of the CA3 region [4, 5] and the intermediate hippocampus [6] in returning to goal locations that the animal has seen only once. This poses the question of whether and how hippocampal dynamics could support a representation of the current location, a representation of a goal, and the relation between the two.

In this article, we propose that a model of CA3 as a "bump attractor" [7] can be be used for path planning. The attractor map represents not only locations within the environment, but also the spatial relationship between locations. In particular, broad activity profiles (like those found in intermediate and ventral hippocampus [8]) can be viewed as a condensed map of a particular environment. The planned path presents as rapid sequential activity from the current position to the goal location, similar to the "preplay" observed experimentally in hippocampal activity during navigation tasks [9, 10], including paths that require navigating around obstacles. In the model, the activity is produced by supplying input to the network consistent with the sensory input that would be provided at the

goal site. Unlike other recent models of rapid goal learning and path planning [11, 12], there is no backwards diffusion of a value signal from the goal to the current state during the learning or planning process. Instead, the sequential activity results from the representation of space in the attractor network, even in the presence of obstacles.

The recurrent structure in our model is derived from the "successor representation" [13], which represents space according to the number and length of paths connecting different locations. The resulting network can be interpreted as an attractor manifold in a low–dimensional space, where the dimensions correspond to weighted version of the most relevant eigenvectors of the environment's transition matrix. Such low–frequency functions have recently found support as a viable basis for place cell activity [14–16]. We show that, when the attractor network operates in this basis and is stimulated with a goal location, the network activity traces out a path to that goal. Thus, the bump attractor network can act as a spatial path planning system as well as a spatial memory system.

## 2 The successor representation and path–finding

A key problem in reinforcement learning is assessing the value of a particular state, given the expected returns from that state in both the immediate and distant future. Several model–free algorithms exist for solving this task [17], but they are slow to adjust when the reward landscape is rapidly changing. The successor representation, proposed by Dayan [13], addresses this issue.

Given a Markov chain described by the transition matrix $\mathbf{P}$, where each element $P(s, s')$ gives the probability of transitioning from state $s$ to state $s'$ in a single time step; a reward vector $\mathbf{r}$, where each element $r(s')$ gives the expected immediate returns from state $s'$; and a discount factor $\gamma$, the expected returns $\mathbf{v}$ from each state can be described by

$$
\begin{aligned}
\mathbf{v} &= \mathbf{r} + \gamma\mathbf{P}\mathbf{r} + \gamma^2\mathbf{P}^2\mathbf{r} + \gamma^3\mathbf{P}^3\mathbf{r} + \dots \\
&= (\mathbf{I} - \gamma\mathbf{P})^{-1}\mathbf{r} \\
&= \mathbf{L}\mathbf{r}.
\end{aligned}
\tag{1}
$$

The successor representation $\mathbf{L}$ provides an efficient means of representing the state space according to the expected (discounted) future occupancy of each state $s'$, given that the chain is initialized from state $s$. An agent employing a policy described by the matrix $\mathbf{P}$ can immediately update the value function when the reward landscape $\mathbf{r}$ changes, without any further exploration.

The successor representation is particularly useful for representing many reward landscapes in the same state space. Here we consider the set of reward functions where returns are confined to a single state $s'$; i.e. $r(s') = \delta_{s'g}$ where $\delta$ denotes the Kronecker delta function and the index $g$ denotes a particular goal state. From Eq. 1, we see that the value function is then given by the column $s'$ of the matrix $\mathbf{L}$. Indeed, when we consider only a single goal, we can see the elements of $\mathbf{L}$ as $L(s, s') = v(s|s' = g)$. We will use this property to generate a spatial mapping that allows for a rapid approximation of the shortest path between any two points in an environment.

### 2.1 Representing space using the successor representation

In the spatial navigation problems considered here, we assume that the animal has explored the environment sufficiently to learn its natural topology. We represent the relationship between locations with a Gaussian affinity metric $a$: given states $s(x, y)$ and $s'(x, y)$ in the 2D plane, their affinity is

$$
a(s(x, y), s'(x, y)) = a(s'(x, y), s(x, y)) = \exp\left(\frac{-d^2}{2\sigma_s^2}\right)
\tag{2}
$$

where $d$ is the length of the shortest traversable path between $s$ and $s'$, respecting walls and obstacles. We define $\sigma$ to be small enough that the metric is localized (Fig. 1) such that $a(s(x, y), \cdot)$ resembles a small bump in space, truncated by walls. Normalizing the affinity metric gives

$$
p(s, s') = \frac{a(s, s')}{\sum_{s'} a(s, s')}.
\tag{3}
$$

The normalized metric can be interpreted as a transition probability for an agent exploring the environment randomly. In this case, a spectral analysis of the successor representation [14, 18] gives

$$v(s|s' = g) = \pi(s') \sum_{l=0}^{n} (1 - \gamma \lambda_l)^{-1} \psi_l(s) \psi_l(s') \tag{4}$$

where $\psi_l$ are the right eigenvectors of the transition matrix $\mathbf{P}$, $1 = |\lambda_0| \geq |\lambda_1| \geq |\lambda_2| \cdots \geq |\lambda_n|$ are the eigenvalues [18], and $\pi(s')$ denotes the steady–state occupancy of state $s'$ resulting from $\mathbf{P}$. Although the affinity metric is defined locally, large–scale features of the environment are represented in the eigenvectors associated with the largest eigenvalues (Fig. 1).

We now express the position in the 2D space using a set of "successor coordinates", such that

$$s(x, y) \mapsto \check{\mathbf{s}} = \left( \sqrt{(1 - \gamma \lambda_0)^{-1}} \psi_0(s), \sqrt{(1 - \gamma \lambda_1)^{-1}} \psi_1(s), \ldots, \sqrt{(1 - \gamma \lambda_q)^{-1}} \psi_q(s) \right) \tag{5}$$

$$= (\xi_0(s), \xi_1(s), \ldots, \xi_q(s))$$

where $\xi_l = \sqrt{(1 - \gamma \lambda_l)^{-1}} \psi_l$. This is similar to the "diffusion map" framework by Coifman and Lafon [18]; with the useful property that, if $q = n$, the value of a given state when considering a given goal is proportional to the scalar product of their respective mappings: $v(s|s' = g) = \pi(s')\langle \check{\mathbf{s}}, \check{\mathbf{s}}' \rangle$. We will use this property to show how a network operating in the successor coordinate space can rapidly generate prospective trajectories between arbitrary locations.

Note that the mapping can also be defined using the eigenvectors $\phi_l$ of a related measure of the space, the normalized graph Laplacian [19]. The eigenvectors $\phi_l$ serve as the objective functions for slow feature analysis [20], and approximations have been extracted through hierarchical slow feature analysis on visual data [15, 16], where they have been used to generate place cell–like behaviour.

## 2.2 Path–finding using the successor coordinate mapping

Successor coordinates provide a means of mapping a set of locations in a 2D environment to a new space based on the topology of the environment. In the new representation, the value landscape is particularly simple. To move from a location $\check{\mathbf{s}}$ towards a goal position $\check{\mathbf{s}}'$, we can consider a constrained gradient ascent procedure on the value landscape:

$$\check{\mathbf{s}}_{t+1} = \underset{\check{\mathbf{s}} \in \check{S}}{\arg\min} \left[ (\check{\mathbf{s}} - (\check{\mathbf{s}}_t + \alpha \nabla v(\check{\mathbf{s}}_t)))^2 \right] \tag{6}$$

$$= \underset{\check{\mathbf{s}} \in \check{S}}{\arg\min} \left[ (\check{\mathbf{s}} - (\check{\mathbf{s}}_t + \tilde{\alpha} \check{\mathbf{s}}'))^2 \right]$$

where $\pi(s')$ has been absorbed into the parameter $\tilde{\alpha}$. At each time step, the state closest to an incremental ascent of the value gradient is selected amongst all states in the environment $\check{S}$. In the following, we will consider how the step $\check{\mathbf{s}}_t + \tilde{\alpha} \check{\mathbf{s}}'$ can be approximated by a neural attractor network acting in successor coordinate space.

Due to the properties of the transition matrix, $\psi_0$ is constant across the state space and does not contribute to the value gradient in Eq. 6. As such, we substituted a free parameter for the coefficient $\sqrt{(1 - \gamma \lambda_0)^{-1}}$, which controlled the overall level of activity in the network simulations.

## 3 Encoding successor coordinates in an attractor network

The bump attractor network is a common model of place cell activity in the hippocampus [7, 21]. Neurons in the attractor network strongly excite other neurons with similar place field centers, and weakly inhibit the neurons within the network with distant place field centers. As a result, the network allows a stable bump of activity to form at an arbitrary location within the environment.

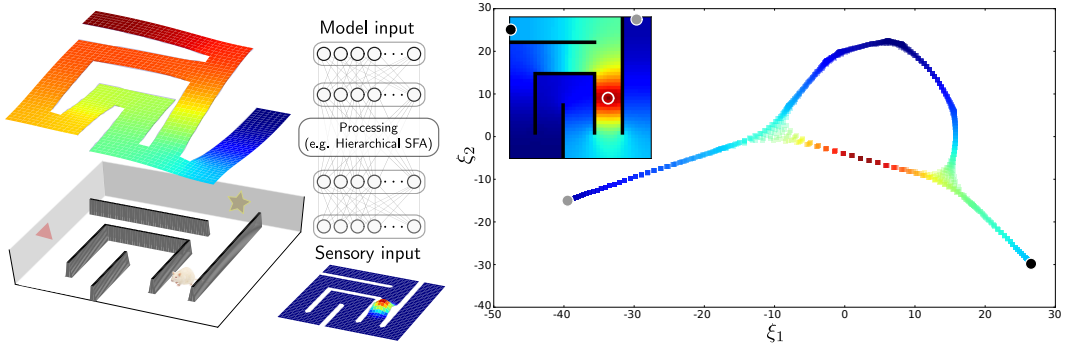

Figure 1: **[Left]** A rat explores a maze–like environment and passively learns its topology. We assume a process such as hierarchical slow feature analysis, that preliminarily extracts slowly changing functions in the environment (here, the vectors $\xi_1 \ldots \xi_q$). The vector $\xi_1$ for the maze is shown in the top left. In practice, we extracted the vectors directly from a localized Gaussian transition function (bottom center, for an arbitrary location). **[Right]** This basis can be used to generate a value map approximation over the environment for a given reward (goal) position and discount factor $\gamma$ (inset). Due to the walls, the function is highly discontinuous in the $xy$ spatial dimensions. The goal position is circled in white. In the scatter plot, the same array of states and value function are shown in the first two non–trivial successor coordinate dimensions. In this space, the value function is proportional to the scalar product between the states and the goal location. The grey and black dots show corresponding states between the inset and the scatter plot.

Such networks typically represent a periodic (toroidal) environment [7, 21], using a local excitatory weight profile that falls off exponentially. Here, we show how the spatial mapping of Eq. 5 can be used to represent bounded environments with arbitrary obstacles. The resulting recurrent weights induce stable firing fields that decrease with distance from the place field center, around walls and obstacles, in a manner consistent with experimental observations [22]. In addition, the network dynamics can be used to perform rapid path planning in the environment.

We will use the techniques introduced in the attractor network models by Eliasmith and Anderson [23] to generalize the bump attractor. We first consider a purely feed–forward network, composed of a population of neurons with place field centers scattered randomly throughout the environment. We assume that the input is highly preprocessed, potentially by several layers of neuronal processing (Fig. 1), and given directly by units $k$ whose activities $\breve{s}_k^{in}(t) = \xi_k(s^{in}(t))$ represent the input in the successor coordinate dimensions introduced above. The activity $a_i$ of neuron $i$ in response to the $m$ inputs $\breve{s}_k^{in}(t)$ can be described by

$$\tau \frac{da_i(t)}{dt} = -a_i(t) + g \left[ \sum_{k=1}^{m} w_{ik}^{ff} \breve{s}_k^{in}(t) \right]_+ \tag{7}$$

where $g$ is a gain factor, $[\cdot]_+$ represents a rectified linear function, and $w_{ik}^{ff}$ are the feed–forward weights. Each neuron is particularly responsive to a "bump" in the environment given by its encoding vector $\mathbf{e_i} = \frac{\breve{s}_i}{||\breve{s}_i||}$, the normalized successor coordinates of a particular point in space, which corresponds to its place field center. The input to neuron $i$ in the network is then given by

$$w_{ik}^{ff} = [\mathbf{e}_i]_k,$$
$$\sum_{k=1}^{m} w_{ik}^{ff} \breve{s}_k^{in}(t) = \mathbf{e}_i \cdot \breve{\mathbf{s}}^{in}(t). \tag{8}$$

A neuron is therefore maximally active when the input coordinates are nearly parallel to its encoding vector. Although we assume the input is given directly in the basis vectors $\xi_l$ for convenience, a neural encoding using an (over)complete basis based on a linear combination of the eigenvectors $\psi_l$ or $\phi_l$ is also possible given a corresponding transformation in the feed–forward weights.

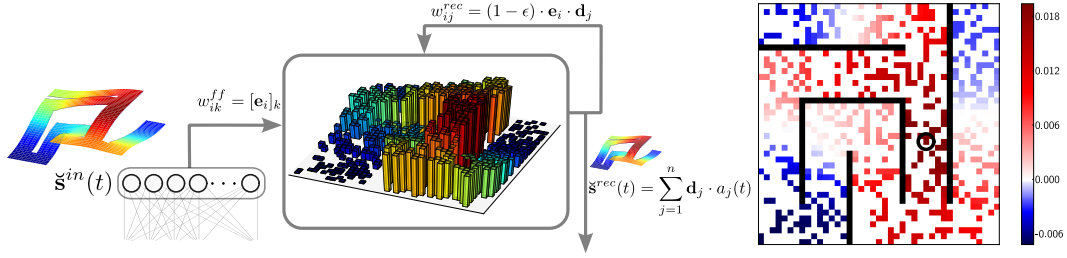

Figure 2: **[Left]** The attractor network structure for the maze–like environment in Fig. 1. The inputs give a low–dimensional approximation of the successor coordinates of a point in space. The network is composed of 500 neurons with encoding vectors representing states scattered randomly throughout the environment. Each neuron's activation is proportional to the scalar product of its encoding vector and the input, resulting in a large "bump" of activity. Recurrent weights are generated using a least–squares error decoding of the successor coordinates from the neural activities, projected back on to the neural encoding vectors. **[Right]** The generated recurrent weights for the network. The plot shows the incoming weights from each neuron to the unit at the circled position, where neurons are plotted according to their place field centers.

If the input $\breve{\mathbf{s}}^{in}(t)$ represents a location in the environment, a bump of activity forms in the network (Fig. 2). These activities give a (non–linear) encoding of the input. Given the response properties of the neurons, we can find a set of linear decoding weights $\mathbf{d}_j$ that recovers an approximation of the input given to the network from the neural activities [23]:

$$\breve{\mathbf{s}}^{rec}(t) = \sum_{j=1}^{n} \mathbf{d}_j \cdot a_j(t). \tag{9}$$

These decoding weights $\mathbf{d}_j$ were derived by minimizing the least–squares estimation error of a set of example inputs from their resulting steady–state activities, where the example inputs correspond to the successor coordinates of points evenly spaced throughout the environment. The minimization can be performed by taking the Moore–Penrose pseudoinverse of the matrix of neural activities in response to the example inputs (with singular values below a certain tolerance removed to avoid overfitting). The vector $\mathbf{d}_j$ therefore gives the contribution of $a_j(t)$ to a linear population code for the input location.

We now introduce the recurrent weights $w_{ij}^{rec}$ to allow the network to maintain a memory of past input in persistent activity. The recurrent weights are determined by projecting the decoded location back on to the neuron encoding vectors such that

$$w_{ij}^{rec} = (1 - \epsilon) \cdot \mathbf{e_i} \cdot \mathbf{d_j}, \tag{10}$$

$$\sum_{j=1}^{n} w_{ij}^{rec} a_j(t) = (1 - \epsilon) \cdot \mathbf{e_i} \cdot \breve{\mathbf{s}}^{rec}(t).$$

Here, the factor $\epsilon \ll 1$ determines the timescale on which the network activity fades. Since the encoding and decoding vectors for the same neuron tend to be similar, recurrent weights are highest between neurons representing similar successor coordinates, and the weight profile decreases with the distance between place field centers (Fig. 2). The full neuron–level description is given by

$$\tau \frac{da_i(t)}{dt} = -a_i(t) + g \left[ \sum_{j=1}^{n} w_{ij}^{rec} a_j(t) + \alpha \sum_{k=1}^{m} w_{ik}^{ff} \breve{s}_k^{in}(t) \right]_+ \tag{11}$$
$$= -a_i(t) + g \left[ \mathbf{e_i} \cdot \left( (1 - \epsilon) \cdot \breve{\mathbf{s}}^{rec}(t) + \alpha \cdot \breve{\mathbf{s}}^{in}(t) \right) \right]_+$$

where the $\alpha$ parameter corresponds to the input strength. If we consider the estimate of $\breve{s}^{rec}(t)$ recovered from decoding the activities of the network, we arrive at the update equation

$$\tau\frac{d\breve{s}^{rec}(t)}{dt} \approx \alpha \cdot \breve{s}^{in}(t) - \epsilon \cdot \breve{s}^{rec}(t). \tag{12}$$

Given a location $\breve{s}^{in}(t)$ as an initial input, the recovered representation $\breve{s}^{rec}(t)$ approximates the input and reinforces it, allowing a persistent bump of activity to form. When $\breve{s}^{in}(t)$ then changes to a new (goal) location, the input and recovered coordinates conflict. By Eq. 12, the recovered location moves in the direction of the new input, giving us an approximation of the initial gradient ascent step in Eq. 6 with the addition of a decay controlled by $\epsilon$. As we will show, the attractor dynamics typically cause the network activity to manifest as a movement of the bump towards the goal location, through locations intermediate to the starting position and the goal (as observed in experiments [9, 10]). After a short stimulation period, the network activity can be decoded to give a state nearby the starting position that is closer to the goal. Note that, with no decay $\epsilon$, the network activity will tend to grow over time. To induce stable activity when the network representation matches the goal position ($\breve{s}^{rec}(t) \approx \breve{s}^{in}(t)$), we balanced the decay and input strength ($\epsilon = \alpha$).

In the following, we consider networks where the successor coordinate representation was truncated to the first $q$ dimensions, where $q \ll n$. This was done because the network is composed of a limited number of neurons, representing only the portion of the successor coordinate space corresponding to actual locations in the environment. In a very high–dimensional space, the network can rapidly move into a regime far from any actual locations, and the integration accuracy suffers. In effect, the weight profiles and feed–forward activation profile become very narrow, and as a result the bump of activity simply disappears from the original position and reappears at the goal. Conversely, low–dimensional representations tend to result in broad excitatory weight profiles and activity profiles (Fig. 2). The high degree of excitatory overlap across the network causes the activity profile to move smoothly between distant points, as we will show.

## 4   Results

We generated attractor networks according to the layout of multiple environments containing walls and obstacles, and stimulated them successively with arbitrary startpoints and goals. We used $n = 500$ neurons to represent each environment, with place field centers selected randomly throughout the environment. The successor coordinates were generated using $\gamma = 1$. We adjusted $q$ to control the dimensionality of the representation. The network activity resembles a bump across a portion of the environment (Fig. 3). Low–dimensional representations (low $q$) produced large activity bumps across significant portions of the environment; when a weak stimulus was provided at the goal, the overall activity decreased while the center of the bump moved towards the goal through the intervening areas of the environment. With a high–dimensional representation, activity bumps became more localized, and shifted discontinuously to the goal (Fig. 3, bottom row).

For several networks representing different environments, we initialized the activity at points evenly spaced throughout the environment and provided weak feed–forward stimulation corresponding to a fixed goal location (Fig. 4). After a short delay ($5\tau$), we decoded the successor coordinates from the network activity to determine the closest state (Eq. 6). The shifts in the network representation are shown by the arrows in Fig. 4. For two networks, we show the effect of different feed–forward stimuli representing different goal locations. The movement of the activity profile was similar to the shortest path towards the goal (Fig. 4, bottom left), including reversals at equidistant points (center bottom of the maze). Irregularities were still present, however, particularly near the edges of the environment and in the immediate vicinity of the goal (where high–frequency components play a larger role in determining the value gradient).

## 5   Discussion

We have presented a spatial bump attractor model generalized to represent environments with arbitrary obstacles, and shown how, with large activity profiles relative to the size of the environment, the network dynamics can be used for path–finding. This provides a possible correlate for goal–directed

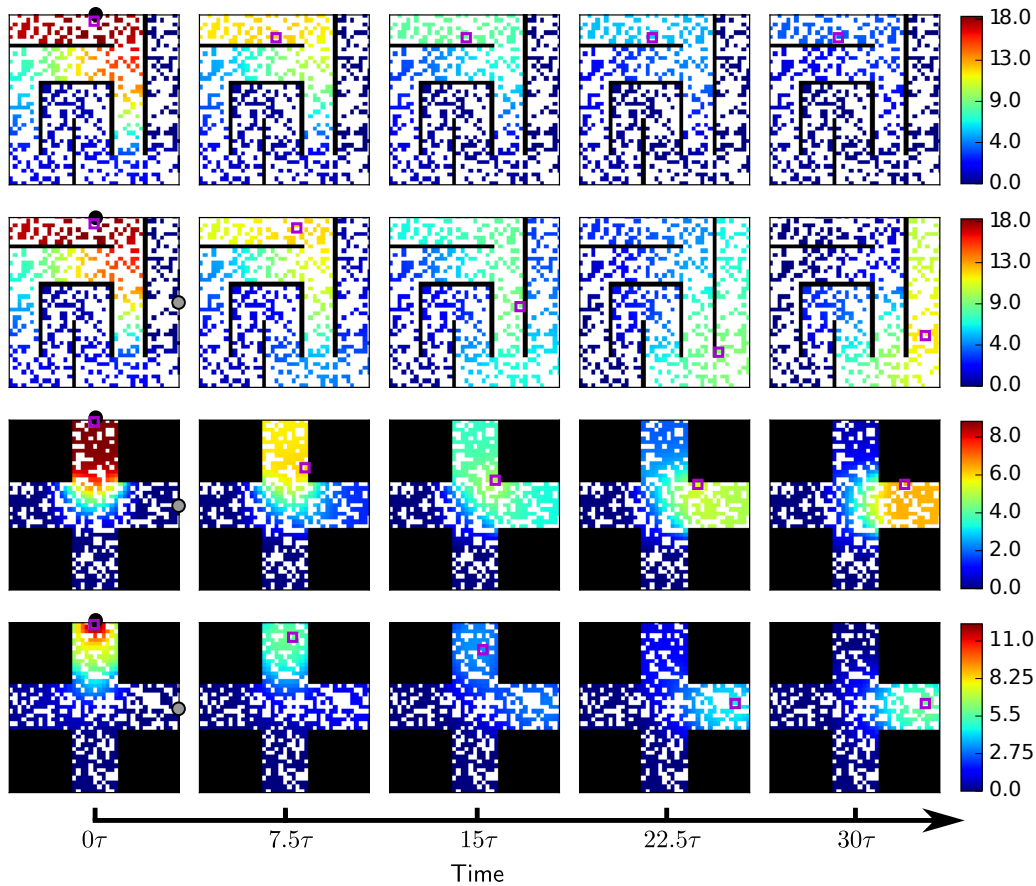

Figure 3: Attractor network activities illustrated over time for different inputs and networks, in multiples of the membrane time constant $\tau$. Purple boxes indicate the most active unit at each point in time. **[First row]** Activities are shown for a network representing a maze–like environment in a low–dimensional space ($q = 5$). The network was initially stimulated with a bump of activation representing the successor coordinates of the state at the black circle; recurrent connections maintain a similar yet fading profile over time. **[Second row]** For the same network and initial conditions, a weak constant stimulus was provided representing the successor coordinates at the grey circle; the activities transiently decrease and the center of the profile shifts over time through the environment. **[Third row]** Two positions (black and grey circles) were sequentially activated in a network representing a second environment in a low–dimensional space ($q = 4$). **[Bottom row]** For a higher–dimensional representation ($q = 50$), the activity profile fades rapidly and reappears at the stimulated position.

activity observed in the hippocampus [9, 10] and an hypothesis for the role that the hippocampus and the CA3 region play in rapid goal–directed navigation [4–6], as a complement to an additional (e.g. model–free) system enabling incremental goal learning in unfamiliar environments [4].

Recent theoretical work has linked the bump–like firing behaviour of place cells to an encoding of the environment based on its natural topology, including obstacles [22], and specifically to the successor representation [14]. As well, recent work has proposed that place cell behaviour can be learned by processing visual data using hierarchical slow feature analysis [15, 16], a process which can extract the lowest frequency eigenvectors of the graph Laplacian generated by the environment [20] and therefore provide a potential input for successor representation–based activity. We provide the first link between these theoretical analyses and attractor–based models of CA3.

Slow feature analysis has been proposed as a natural outcome of a plasticity rule based on Spike–Timing–Dependent Plasticity (STDP) [24], albeit on the timescale of a standard postsynaptic po-

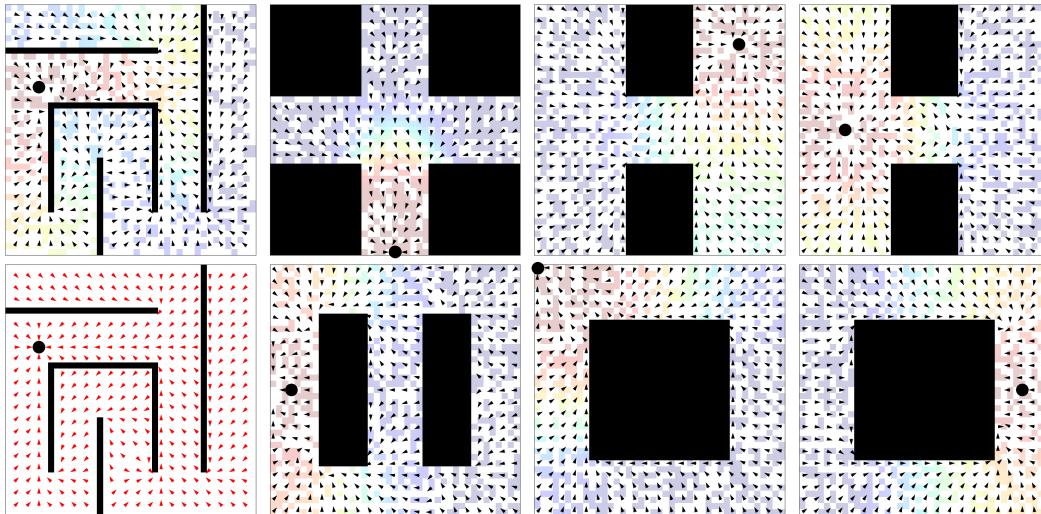

Figure 4: Large–scale, low–dimensional attractor network activities can be decoded to determine local trajectories to long–distance goals. Arrows show the initial change in the location of the activity profile by determining the state closest to the decoded network activity (at $t = 5\tau$) after weakly stimulating with the successor coordinates at the black dot ($\alpha = \epsilon = 0.05$). Pixels show the place field centers of the 500 neurons representing each environment, coloured according to their activity at the stimulated goal site. **[Top left]** Change in position of the activity profile in a maze–like environment with low–dimensional activity ($q = 5$) compared to **[Bottom left]** the true shortest path towards the goal at each point in the environment. **[Additional plots]** Various environments and stimulated goal sites using low–dimensional successor coordinate representations.

tential rather than the behavioural timescale we consider here. However, STDP can be extended to behavioural timescales when combined with sustained firing and slowly decaying potentials [25] of the type observed on the single–neuron level in the input pathway to CA3 [26], or as a result of network effects. Within the attractor network, learning could potentially be addressed by a rule that trains recurrent synapses to reproduce feed–forward inputs during exploration (e.g. [27]).

Our model assigns a key role to neurons with large place fields in generating long–distance goal–directed trajectories. This suggests that such trajectories in dorsal hippocampus (where place fields are much smaller [8]) must be inherited from dynamics in ventral or intermediate hippocampus. The model predicts that ablating the intermediate/ventral hippocampus [6] will result in a significant reduction in goal–directed preplay activity in the remaining dorsal region. In an intact hippocampus, the model predicts that long–distance goal–directed preplay in the dorsal hippocampus is preceded by preplay tracing a similar path in intermediate hippocampus. However, these large–scale networks lack the specificity to consistently generate useful trajectories in the immediate vicinity of the goal. Therefore, higher–dimensional (dorsal) representations may prove useful in generating trajectories close to the goal location, or alternative methods of navigation may become more important.

If an assembly of neurons projecting to the attractor network is active while the animal searches the environment, reward–modulated Hebbian plasticity provides a potential mechanism for reactivating a goal location. In particular, the presence of a reward–induced neuromodulator could allow for potentiation between the assembly and the attractor network neurons active when the animal receives a reward at a particular location. Activating the assembly would then provide stimulation to the goal location in the network; the same mechanism could allow an arbitrary number of assemblies to become selective for different goal locations in the same environment. Unlike traditional model–free methods of learning which generate a static value map, this would give a highly configurable means of navigating the environment (e.g. visiting different goal locations based on thirst vs. hunger needs), providing a link between spatial navigation and higher cognitive functioning.

**Acknowledgements**
This research was supported by the Swiss National Science Foundation (grant agreement no. 200020_147200). We thank Laureline Logiaco and Johanni Brea for valuable discussions.

# References

[1] William Beecher Scoville and Brenda Milner. Loss of recent memory after bilateral hippocampal lesions. *Journal of neurology, neurosurgery, and psychiatry*, 20(1):11, 1957.

[2] Howard Eichenbaum. *Memory, amnesia, and the hippocampal system*. MIT press, 1993.

[3] John O'Keefe and Jonathan Dostrovsky. The hippocampus as a spatial map. preliminary evidence from unit activity in the freely-moving rat. *Brain research*, 34(1):171–175, 1971.

[4] Kazu Nakazawa, Linus D Sun, Michael C Quirk, Laure Rondi-Reig, Matthew A Wilson, and Susumu Tonegawa. Hippocampal CA3 NMDA receptors are crucial for memory acquisition of one-time experience. *Neuron*, 38(2):305–315, 2003.

[5] Toshiaki Nakashiba, Jennie Z Young, Thomas J McHugh, Derek L Buhl, and Susumu Tonegawa. Transgenic inhibition of synaptic transmission reveals role of ca3 output in hippocampal learning. *Science*, 319(5867):1260–1264, 2008.

[6] Tobias Bast, Iain A Wilson, Menno P Witter, and Richard GM Morris. From rapid place learning to behavioral performance: a key role for the intermediate hippocampus. *PLoS biology*, 7(4):e1000089, 2009.

[7] Alexei Samsonovich and Bruce L McNaughton. Path integration and cognitive mapping in a continuous attractor neural network model. *The Journal of Neuroscience*, 17(15):5900–5920, 1997.

[8] Kirsten Brun Kjelstrup, Trygve Solstad, Vegard Heimly Brun, Torkel Hafting, Stefan Leutgeb, Menno P Witter, Edvard I Moser, and May-Britt Moser. Finite scale of spatial representation in the hippocampus. *Science*, 321(5885):140–143, 2008.

[9] Brad E Pfeiffer and David J Foster. Hippocampal place-cell sequences depict future paths to remembered goals. *Nature*, 497(7447):74–79, 2013.

[10] Andrew M Wikenheiser and A David Redish. Hippocampal theta sequences reflect current goals. *Nature neuroscience*, 2015.

[11] Louis-Emmanuel Martinet, Denis Sheynikhovich, Karim Benchenane, and Angelo Arleo. Spatial learning and action planning in a prefrontal cortical network model. *PLoS computational biology*, 7(5):e1002045, 2011.

[12] Filip Ponulak and John J Hopfield. Rapid, parallel path planning by propagating wavefronts of spiking neural activity. *Frontiers in computational neuroscience*, 7, 2013.

[13] Peter Dayan. Improving generalization for temporal difference learning: The successor representation. *Neural Computation*, 5(4):613–624, 1993.

[14] Kimberly L Stachenfeld, Matthew Botvinick, and Samuel J Gershman. Design principles of the hippocampal cognitive map. In Z. Ghahramani, M. Welling, C. Cortes, N.D. Lawrence, and K.Q. Weinberger, editors, *Advances in Neural Information Processing Systems 27*, pages 2528–2536. Curran Associates, Inc., 2014.

[15] Mathias Franzius, Henning Sprekeler, and Laurenz Wiskott. Slowness and sparseness lead to place, head-direction, and spatial-view cells. *PLoS Computational Biology*, 3(8):e166, 2007.

[16] Fabian Schoenfeld and Laurenz Wiskott. Modeling place field activity with hierarchical slow feature analysis. *Frontiers in Computational Neuroscience*, 9:51, 2015.

[17] Richard S Sutton and Andrew G Barto. *Introduction to reinforcement learning*. MIT Press, 1998.

[18] Ronald R Coifman and Stéphane Lafon. Diffusion maps. *Applied and computational harmonic analysis*, 21(1):5–30, 2006.

[19] Sridhar Mahadevan. *Learning Representation and Control in Markov Decision Processes*, volume 3. Now Publishers Inc, 2009.

[20] Henning Sprekeler. On the relation of slow feature analysis and laplacian eigenmaps. *Neural computation*, 23(12):3287–3302, 2011.

[21] John Conklin and Chris Eliasmith. A controlled attractor network model of path integration in the rat. *Journal of computational neuroscience*, 18(2):183–203, 2005.

[22] Nicholas J Gustafson and Nathaniel D Daw. Grid cells, place cells, and geodesic generalization for spatial reinforcement learning. *PLoS computational biology*, 7(10):e1002235, 2011.

[23] Chris Eliasmith and C Charles H Anderson. *Neural engineering: Computation, representation, and dynamics in neurobiological systems*. MIT Press, 2004.

[24] Henning Sprekeler, Christian Michaelis, and Laurenz Wiskott. Slowness: an objective for spike-timing-dependent plasticity. *PLoS Comput Biol*, 3(6):e112, 2007.

[25] Patrick J Drew and LF Abbott. Extending the effects of spike-timing-dependent plasticity to behavioral timescales. *Proceedings of the National Academy of Sciences*, 103(23):8876–8881, 2006.

[26] Phillip Larimer and Ben W Strowbridge. Representing information in cell assemblies: persistent activity mediated by semilunar granule cells. *Nature neuroscience*, 13(2):213–222, 2010.

[27] Robert Urbanczik and Walter Senn. Learning by the dendritic prediction of somatic spiking. *Neuron*, 81(3):521–528, 2014.

